# Receptive field structure of flow detectors for heading perception

**Jaap A. Beintema**
Dept. Zoology & Neurobiology
Ruhr University Bochum, Germany, 44780
*beintema@neurobiologie.ruhr-uni-bochum.de*

**Albert V. van den Berg**
Dept. of Neuro-ethology, Helmholtz Institute,
Utrecht University, The Netherlands
*a.v.vandenberg@bio.uu.nl*

**Markus Lappe**
Dept. Zoology & Neurobiology
Ruhr University Bochum, Germany, 44780
*lappe@neurobiologie.ruhr-uni-bochum.de*

## Abstract

Observer translation relative to the world creates image flow that expands from the observer's direction of translation (heading) from which the observer can recover heading direction. Yet, the image flow is often more complex, depending on rotation of the eye, scene layout and translation velocity. A number of models [1-4] have been proposed on how the human visual system extracts heading from flow in a neurophysiologically plausible way. These models represent heading by a set of neurons that respond to large image flow patterns and receive input from motion sensed at different image locations. We analysed these models to determine the exact receptive field of these heading detectors. We find most models predict that, contrary to widespread believe, the contributing motion sensors have a preferred motion directed circularly rather than radially around the detector's preferred heading. Moreover, the results suggest to look for more refined structure within the circular flow, such as bi-circularity or local motion-opponency.

## Introduction

The image flow can be considerably more complicated than merely an expanding pattern of motion vectors centered on the heading direction (Fig. 1). Flow caused by eye rotation (Fig. 1b) causes the center of flow to be displaced (compare Fig. 1a and c). The effect of rotation depends on the ratio of rotation and translation speed.

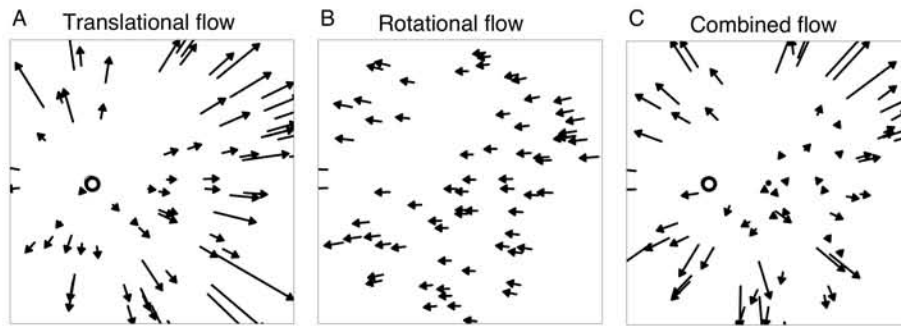

Figure 1: Flow during a) observer translation through a 3D-cloud of dots, headed 10° towards the left, during b) observer rotation about the vertical towards the right, and during c) the combination of both.

Also, since the image motions caused by translation depend on point distance and the image motions caused by rotation do not, the combined movement results in flow that is no longer purely expanding for scenes containing depth differences (Fig. 1c). Heading detection can therefore not rely on a simple extrapolation mechanism that determines the point of intersection of motion vectors.

A number of physiologically-based models [1-4] have been proposed on how the visual system might arrive at a representation of heading from flow that is insensitive to parameters other than heading direction. These models assume heading is encoded by a set of units that each respond best to a specific pattern of flow that matches their preferred heading. Such units resemble neurons found in monkey brain area MST. MST cells have large receptive fields (RF), typically covering one quart or more of the visual field, and receive input from several local motion sensors in brain area MT. The receptive field of MST neurons may thus be defined as the preferred location, speed and direction of all input local motion sensors. Little is known yet about the RF structure of MST neurons. We looked for similarities between current models at the level of the RF structure. First we explain the RF structure of units in the velocity gain model, because this model makes clear assumptions on the RF structure. Next, we we show the results of reconstructing RF structure of units in the population model[2]. Finally, we analyse the RF structure of the template model[3] and motion-opponency model[4].

## Velocity gain field model

The velocity gain field model[1] is based on flow templates. A flow template, as introduced by Perrone and Stone[3], is a unit that evaluates the evidence that the flow fits the unit's preferred flow field by summing the responses of local motion sensors outputs. Heading is then represented by the preferred heading direction of the most active template(s). The velocity gain field model[1] is different from Perrone and Stone's template model[2] in the way it acquires invariance for translation speed, point distances and eye rotation. Whereas the template model requires a different template for each possible combination of heading direction and rotation, the velocity gain field model obtains rotation invariance using far less templates by exploiting eye rotation velocity signals.

The general scheme applied in the velocity gain field model is as follows. In a set of flow templates, each tuned to pure expansion with specific preferred heading,

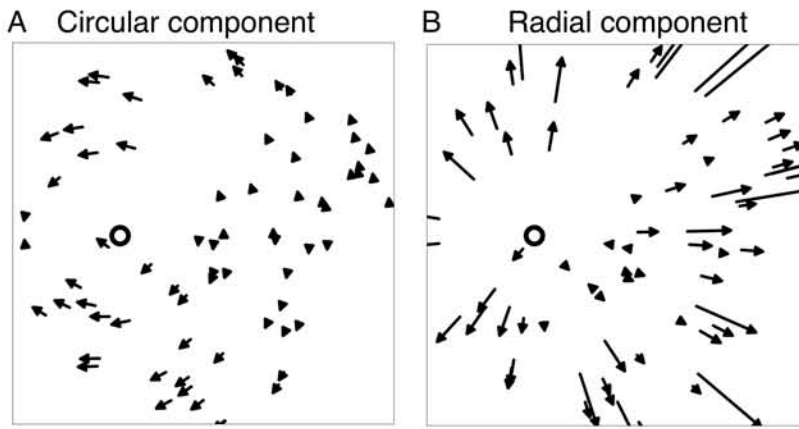

Figure 2: The heading-centered circular (a) and radial (b) component of the flow during combined translation and rotation as in Fig. 2c.

the templates would change their activity during eye rotation. Simply subtracting the rotation velocity signal for each flow template would not suffice to compensate because each template is differently affected by rotational flow. However, each flow template can become approximately rotation-invariant by subtracting a gain field activity that is a multiplication of the eye velocity $\epsilon$ with a derivative template activity $\delta O/\delta R$ that is specific for each flow template. The latter reflects the change in flow template activity $O$ given a change in rotational flow $\delta R$. Such derivative template $\delta O/\delta R$ can be constructed from the activity difference of two templates tuned to the same heading, but opposite rotation. Thus, in the velocity gain field model, templates tuned to heading direction and a component of rotation play an important role.

To further appreciate the idea behind the RF structure in the velocity gain field model, note that the retinal flow can be split into a circular and radial component, centered on the heading point (Fig. 2). Translation at different speeds or through a different 3D environment will alter the radial component only. The circular component contains a rotational component of flow but does not change with point distances or translational speed. This observation lead to the assumption implemented in the velocity gain field model that templates should only measure the flow along circles centered on the point of preferred heading.

An example of the RF structure of a typical unit in the velocity gain field model, tuned to heading and rightward rotation is shown in Fig. 3. This *circular* RF structure strongly reduces sensitivity to variations in depth structure or the translational speed, while the template's tuning to heading direction is preserved, because its preferred structure is centered on its preferred heading direction [1]. Interestingly, the RF structure of the typical rotation-tuned heading units is *bi-circular*, because the direction of circular flow is opponent in the hemifields to either side of an axis (in this case the horizontal axis) through the heading point. Moreover, the structure contains a gradient in magnitude along the circle, decreasing towards the horizontal axis.

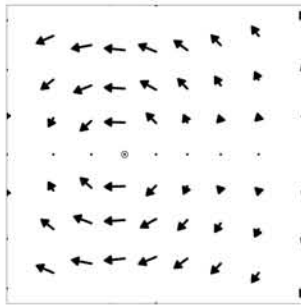

Figure 3: Bi-circular RF structure of a typical unit in the velocity gain field model, tuned to leftward heading and simultaneous rightward rotation about the vertical. Individual vectors show the preferred direction and velocity of the input motion sensors.

## Population model

The population model [2] derives a representation of heading direction that is invariant to the other flow parameters using a totally different approach. This model does not presume an explicit RF structure. Instead, the connections strengths and preferred directions of local motion inputs to heading-specific flow units are computed according to an optimizing algorithm[5]. We here present the results obtained for a restricted version of the model in which eye rotation is assumed to be limited to pursuit that keeps the eye fixated on a stationary point in the scene during the observer translation. Specifically, we investigated whether a circular or bi-circular RF structure as predicted by the velocity gain model emerges in the population model.

The population model [2,6] is an implementation of the subspace algorithm by Heeger and Jepson [5] into a neural network. The subspace algorithm computes a residual function $R(\mathbf{T}_j)$ for a range of possible preferred heading directions. The residual function is minimized when flow vectors measured at $m$ image locations, described as one array, are perpendicular to the vectors that form columns of a matrix $\mathbf{C}^\perp(\mathbf{T}_j)$. This matrix is computed from the preferred 3-D translation vector $\mathbf{T}_j$ and the $m$ image locations. Thus, by finding the matrix that minimizes the residue, the algorithm has solved the heading, irrespective of the 3D-rotation vector, unknown depths of points and translation speed.

To implement the subspace algorithm in a neurophysiologically plausible way, the population model assumes two layers of units. The first MT-like layer contains local motion sensors that fire linearly with speed and have cosine-like direction tuning. These sensors connect to units in the second MST-like layer. The activity in a 2nd layer unit, with specific preferred heading $\mathbf{T}_j$, represents the likelihood that the residual function is zero. The connection strengths are determined by the $\mathbf{C}^\perp(\mathbf{T}_j)$ matrix. As not to have too many motion inputs per 2nd layer unit, the residual function $R(\mathbf{T}_j)$ is partitioned into smaller subresidues that take only a few motion inputs. The likelihood for a specific heading is then given by the sum of responses in a population with same preferred heading.

Given the image locations and the preferred heading, one can reconstruct the RF structure for 2nd layer units with the same preferred heading. The preferred motion inputs to a second layer unit are given by vectors that make up each column of $\mathbf{C}^\perp(\mathbf{T}_j)$. Hereby, the vector direction represents the preferred motion direction,

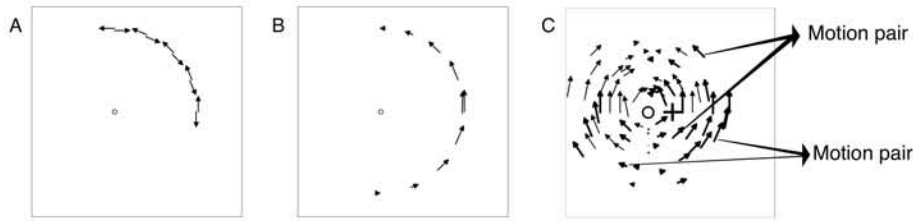

Figure 4: Examples of receptive field structure of a population that encodes heading $10°$ towards the left (circle). a-b) Five pairs of MT-like sensors, where the motion sensors of each pair are at a) the same image location, or b) at image locations one quarter of a cycle apart. c) Distribution of multiple pairs leading to bi-circular pattern.

and the vector magnitude represents the strength of the synaptic connection. The matrix $\mathbf{C}^{\perp}(\mathbf{T}_j)$ is computed from the orthogonal complement of a $(2m \times m + 3)$ matrix $\mathbf{C}(\mathbf{T}_j)$ [5]. On the assumption that only fixational eye movements occur, the matrix reduces to $(2m \times m + 1)$[6]. Given only two flow vector inputs $(m = 2)$, the matrix $\mathbf{C}^{\perp}(\mathbf{T}_j)$ reduces to one column of length $m = 4$. The orthogonal complement of this $4 \times 3$ matrix was solved in Mathematica by first computing the nullspace of the inverse matrix of $\mathbf{C}(\mathbf{T}_j)$, and then constructing an orthonormal basis for it using Gram-Schmidt orthogonalisation. We computed the orientation and magnitude of the two MT-inputs analytically. Instead of giving the mathematics, we here describe the main results.

### Circularity

Independent of the spatial arrangement of the two MT-inputs to a 2nd-layer unit, their preferred motions turned out to be always directed along a circle centered on the preferred heading point. Fig. 4 shows examples of the circular RF structures, for different distributions of motion pairs that code for the same heading direction.

### Motion-opponency

For pairs of motion sensors at overlapping locations, the vectors of each pair always turned out to be opponent and of equal magnitude (Fig. 4a). For pairs of spatially separated motion sensors, the preferred magnitude and direction of the two motion inputs depend on their location with respect to the hemispheres divided by the line through heading and fixation point. We find that preferred motion directions are opponent if the pair is located within the same hemifield, but uni-directional if the pair is split across the two hemifields as in Fig. 4b.

### Bi-circularity

Interestingly, if pairs of motion sensors are split across hemi fields, with partners at image locations $90°$ rotated about the heading point, a magnitude gradient appears in the RF structure (Fig. 4b). Thus, with these pairs a bi-circular RF structure can be constructed similar to units tuned to rotation about the vertical in the velocity gain field model (compare with Fig. 3).

Note, that the bi-circular RF structures do differ since the axis along which the largest magnitude occurs is horizontal for the population model and vertical for the velocity gain field model. The RF structure of the population model unit resembles a velocity gain field unit tuned to rotation about the horizontal axis, implying a

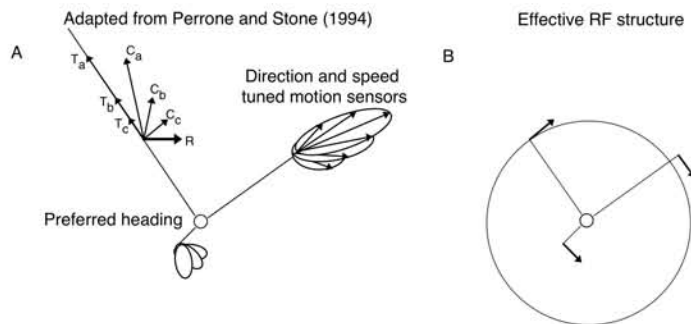

Figure 5: Adapted from Perrone and Stone 1994). a) Each detector sums the responses of the most active sensor at each location. This most active motion sensor is selected from a pool of sensors tuned to different depth planes (Ca, Cb, etc). These vectors are the vector sums of preferred rotation component R and translational components Ta, Tb, etc. b) Effective RF structure.

large sensitivity to such rotation. This, however, does not conflict with the expected performance of the population model. Because in this restricted version rotation invariance is expected only for rotation that keeps the point of interest in the center of the image plane (in this case rotation about the vertical because heading is leftward) units are likely to be sensitive to rotation about the horizontal and torsional axis.

## Template model

The template model and the velocity gain field model differ in how invariance for translation velocities, depth structure and eye rotation is obtained. Here, we investigate whether this difference affects the predicted RF structure. In the template model of Perrone and Stone [3], a template invariant to translation velocity or depth structure is obtained by summing the responses of the most active sensor at each image location. This most active sensor is selected from a collection of motion sensors, each tuned to a different ego-translation speed (or depth plane), but with the same preferred ego-rotation and heading direction (Fig. 5a). Given a large range of depth planes, it follows that a different radial component of motion will stimulate another sensor maximally, but that activity nevertheless remains the same. The contributing response will change only due to a component of motion along a circle centered on the heading, such as is the case when heading direction or rotation is varied. Thus, the contributing response will always be from the motion sensor oriented along the circle around the template's preferred heading. Effectively, this leads to a bi-circular RF structure for units tuned to heading and rotation (Fig. 5b).

## Motion-opponency model

Royden[4] proposed that the effect of rotation is removed at local motion detection level before the motion signals are received by flow detectors. This is achieved by MT-like sensors that compute the difference vector between spatially neighbouring motion vectors. Such difference vector will always be oriented along lines intersecting at the heading point (Fig. 6). Thus, the resulting input to flow detectors will be oriented radially. Indeed, Royden's results[4] show that the preferred directions of the operators with the largest response will be radially, not circularly, oriented.

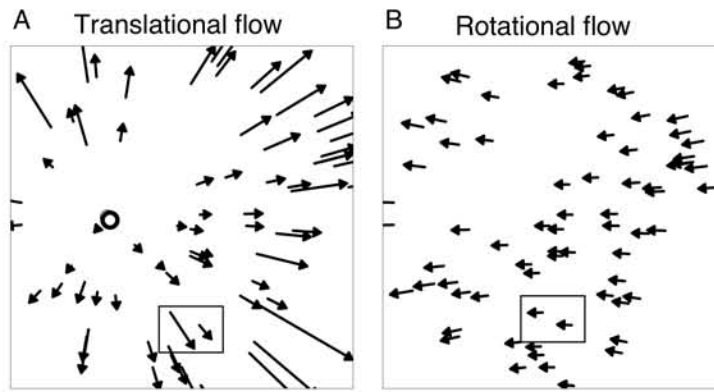

Figure 6: Motion parallax, the difference vector between locally neighbouring motion vectors. For translation flow (a) the difference vector will be oriented along line through the heading point, whereas for rotational flow (b) the difference vector vanishes (compare vectors within square).

## Summary and Discussion

We showed that a circular RF structure, such as proposed by the velocity gain field model[1], is also found in the population model[2] and is effectively present in the template model[3] as well. Only the motion-opponent model [4] prefers radial RF structures. Furthermore, we find that under certain restrictions, the population model reveals local motion-opponency and bi-circularity, properties that can be found in the other models as well.

A circular RF structure turns out to be a prominent property in three models. This supports the counterintuitive, but computationally sensible idea, that it is not the radial flow structure, but the structure perpendicular to it, that contributes to the response of heading-sensitive units in the human brain. Studies on area MST cells not only report selectivity for expanding motion patterns, but also a significant proportion of cells that are selective to rotation patterns [7-10]. These models could explain why cells respond so well to circular motion, in particular to the high rotation speeds (up to about 80 deg/s) not experienced in daily life.

This model study suggests that selectivity for circular flow has a direct link to heading detection mechanisms. It also suggests that testing selectivity for expanding motion might be a bad indicator for determining a cell's preferred heading. This point has been noted before, as MST seems to be systematically tuned to the focus of rotation, exactly like model neurons [9].

Little is still known about the receptive field structure of MST cells. So far the receptive field structure of MST cells has only been roughly probed [10], and the results neither support a radial nor a circular structure. Also, so far only uni-circular motion has been tested. Our analyses points out that it would be worthwhile to look for more refined circular structure such as local motion-opponency. Local motion opponency has already been found in area MT, where some cells respond only if different parts of their receptive field are stimulated with different motion [11]. Another promising structure to look for would be bi-circularity, with gradients in magnitude of preferred motion along the circles.

## Acknowledgments

Supported by the German Science Foundation and the German Federal Ministry of Education and Research.

## References

[1] Beintema, J. A. & van den Berg A. V. (1998) Heading detection using motion templates and eye velocity gain fields. *Vision Research*, **38**(14):2155-2179.

[2] Lappe M., & Rauschecker J. P. (1993) A neural network for the processing of optic flow from ego-motion in man and higher mammals. *Neural Computation*, **5**:374-391.

[3] Perrone J. A. & Stone L. S. (1994) A model for the self-motion estimation within primate extrastriate visual cortex. *Vision Research*, **34**:2917-2938.

[4] Royden C. S. (1997) Mathematical analysis of motion-opponent mechanisms used in the determination of heading and depth. *Journal of the Optical Society of America A*, **14**(9):2128-2143.

[5] Heeger D. J. & Jepson A. D. (1992) Subspace methods for recovering rigid motion I: Algorithm and implementation. *International Journal of Computational Vision*, **7**:95-117.

[6] Lappe M. & Rauschecker J.P. (1993) Computation of heading direction from optic flow in visual cortex. In C.L. Giles, S.J. Hanson and J.D. Cowan (eds.), *Advances in Neural Information Processing Systems 5*, pp. 433-440. Morgan Kaufmann.

[7] Tanaka K. & Saito H. (1989) Analysis of the visual field by direction, expansion/contraction, and rotation cells clustered in the dorsal part of the medial superior temporal area of the macaque monkey *Journal of Neurophysiology*, **62**(3):626-641.

[8] Duffy C. J. & Wurtz R. H. (1991) Sensitivity of MST neurons to optic flow stimuli. I. A continuum of response selectivity to large-field stimuli. *Journal of Neurophysiology*, **65**(6):1329-1345.

[9] Lappe M., Bremmer F., Pekel M., Thiele A., Hoffmann K.-P. (1996) Optic flow processing in monkey STS: a theoretical and experimental approach. *the Journal of Neuroscience*, **16**(19):6265-6285.

[10] Duffy C. J. & Wurtz R. H. (1991) Sensitivity of MST neurons to optic flow stimuli. II. Mechanisms of response selectivity revealed by small-field stimuli. *Journal of Neurophysiology*, **65**(6):1346-1359.

[11] Allman J., Miezin F. & McGuinness E. (1985) Stimulus specific responses from beyond the classical receptive field: Neurophysiological mechanisms for local-global comparisons in visual neurons. *Ann. Rev. Neurosci.*, **8**:407-430.
